# Multilabel Classification using Bayesian Compressed Sensing

**Ashish Kapoor**[†]**, Prateek Jain**[‡] **and Raajay Viswanathan**[‡]
[†]Microsoft Research, Redmond, USA
[‡]Microsoft Research, Bangalore, INDIA
`{akapoor, prajain, t-rviswa}@microsoft.com`

## Abstract

In this paper, we present a Bayesian framework for multilabel classification using compressed sensing. The key idea in compressed sensing for multilabel classification is to first project the label vector to a lower dimensional space using a random transformation and then learn regression functions over these projections. Our approach considers both of these components in a single probabilistic model, thereby jointly optimizing over compression as well as learning tasks. We then derive an efficient variational inference scheme that provides joint posterior distribution over all the unobserved labels. The two key benefits of the model are that a) it can naturally handle datasets that have missing labels and b) it can also measure uncertainty in prediction. The uncertainty estimate provided by the model allows for active learning paradigms where an oracle provides information about labels that promise to be maximally informative for the prediction task. Our experiments show significant boost over prior methods in terms of prediction performance over benchmark datasets, both in the fully labeled and the missing labels case. Finally, we also highlight various useful active learning scenarios that are enabled by the probabilistic model.

## 1 Introduction

Large scale multilabel classification problems arise in several practical applications and has recently generated a lot of interest with several efficient algorithms being proposed for different settings [1, 2]. A primary reason for thrust in this area is due to explosion of web-based applications, such as Picasa, Facebook and other online sharing sites, that can obtain multiple tags per data point. For example, users on the web can annotate videos and images with several possible labels. Such applications have provided a new dimension to the problem as these applications typically have millions of tags. Most of the existing multilabel methods learn a decision function or weight vector *per label* and then combine the decision functions in a certain manner to predict labels for an unseen point [3, 4, 2, 5, 6]. However, such approaches quickly become infeasible in real-world as the number of labels in such applications is typically very large. For instance, traditional multi-label classification techniques based on 1-vs-all SVM [7] is prohibitive because of both large train and test times.

To alleviate this problem, [1] proposed a compressed sensing (CS) based method that exploits the fact that usually the label vectors are very sparse, i.e., the number of positive labels/tags present in a point is significantly less than the total number of labels. Their algorithm uses the following result from the CS literature: an $s$-sparse vector in $\mathbb{R}^L$ can be recovered efficiently using $K = O(s \log L/s)$ measurements. Their method projects label vectors into a $s \log L/s$ dimensional space and learns a regression function in the projected space (independently for each dimension). For test points, the learnt regression function is applied in the reduced space and then standard recovery algorithms from CS literature are used to obtain sparse predicted labels [8, 9]. However, in

this method, learning of the decision functions is independent of the sparse recovery and hence in practice, it requires several measurements to match accuracy of the standard baseline methods such as 1-vs-all SVM. Another limitation of this method is that the scheme does not directly apply when labels are missing, a common aspect in real-world web applications. Finally, the method does not lend itself naturally to uncertainty analysis that can be used for active learning of labels.

In this paper, we address some of the issues mentioned above using a novel Bayesian framework for multilabel classification. In particular, we propose a joint probabilistic model that combines compressed sensing [10, 11] with a Bayesian learning model on the projected space. Our model can be seen as a Bayesian co-training model, where the lower dimensional projected space can be thought of as latent variables. And these latent variables are generated by two different views: a) using a random projection of the label vector, b) using a (linear) predictor over the input data space. Hence, unlike the method of [1], our model can jointly infer predictions in the projected space and projections of the label vector. This joint inference leads to more efficient utilization of the latent variable space and leads to significantly better accuracies than the method of [1] while using same number of latent variables $K$.

Besides better prediction performance, there are several other advantages offered by our probabilistic model. First, the model naturally handles missing labels as the missing labels are modeled as random variables that can be marginalized out. Second, the model enables derivation of a variational inference method that can efficiently compute joint posterior distribution over all the unobserved random variables. Thus, we can infer labels not only for the test point but also for all the missing labels in the training set. Finally, the inferred posterior over labels provide an estimate of uncertainty making the proposed method amenable to active learning.

Active learning is an important learning paradigm that has received a lot of attention due to the availability of large unlabeled data but paucity of labels over these data sets. In the traditional active learning setting (for binary/multiclass classification), at each round the learner actively seeks labels for a selected unlabeled point and updates its models using the provided label. Several criteria, such as uncertainty [12], expected informativeness [13, 14], reduction in version space [15], disagreement among a committee of classifiers [16], etc. have been proposed. While heuristics have been proposed [17] in the case of 1-vs-all SVMs, it is still unclear how these methods can be extended to multilabel classification setting in a principled manner. Our proposed model naturally handles the active learning task as the variational inference procedure provides the required posteriors which can guide information acquisition. Further, besides the traditional active learning scenario, where all the labels are revealed for a selected data, the model leads to extension of information foraging to more practical and novel scenarios. For example, we introduce *active diagnosis*, where the algorithm only asks about labels for the test case that potentially can help with prediction over the rest of the unobserved tags. Similarly, we can extend to a *generalized active learning* setting, where the method seeks answer to questions of the type: "does label 'A' exists in data point x". Such extensions are made feasible due to the Bayesian interpretation of the multilabel classification task.

We demonstrate the above mentioned advantages of our model using empirical validation on benchmark datasets. In particular, experiments show that the method significantly outperforms ML-CS based method by [1] and also obtains accuracies matching 1-vs-all SVM while projecting onto $K$-dimensional space that is typically less than half the total number of labels. We expect these gains to become even more significant for datasets with larger number of labels. We also show that the proposed framework is robust to missing labels and actually outperforms 1-vs-all SVM with about 85-95% missing labels while using $K = .5L$ only. Finally, we demonstrate that our active learning strategies select significantly more informative labels/points than the random selection strategy.

## 2   Approach

Assume that we are given a set of training data points $\mathbf{X} = \{\mathbf{x}_i\}$ with labels $\mathbf{Y} = \{\mathbf{y}_i\}$, where each $\mathbf{y}_i = [y_i^1, .., y_i^L] \in [0, 1]^L$ is a multilabel binary vector of size $L$. Further, let us assume that there are data points in the training set for which we have partially observed labeled vectors that leads to the following partitioning: $\mathbf{X} = \mathbf{X}_{\mathcal{L}} \cup \mathbf{X}_{\mathcal{P}}$. Here the subscripts $\mathcal{L}$ and $\mathcal{P}$ indicate fully and partially labeled data respectively. Our goal then is to correctly predict all the labels for data in the test set $\mathbf{X}_{\mathcal{U}}$. Further, we also seek an active learning procedure that would request as few labels as possible from an oracle to maximize classification rate over the test set.

If we treat each label independently then standard machine learning procedures could be used to train individual classifiers and this can even be extended to do active learning. However, such procedures

can be fairly expensive when the number of labels is huge. Further, these methods would simply ignore the missing data, thus may not utilize statistical relationship amongst the labels. Recent techniques in multilabel classification alleviate the problem of large output space [1, 18], but cannot handle the missing data cases. Finally, there are no clear methods of extending these approaches for active learning.

We present a probabilistic graphical model that builds upon ideas of compressed sensing and utilizes statistical relations across the output space for prediction and active information acquisition. The key idea in compressed sensing is to consider a linear transformation of the $L$ dimensional label vector $\mathbf{y}$ to a $K$ dimensional space $\mathbf{z}$, where $K \ll L$, via a random matrix $\boldsymbol{\Phi}$. The efficiency in the classification system is improved by considering regression functions to the compressed vectors $\mathbf{z}$ instead of the true label space. The proposed framework considers Gaussian process priors over the compressed label space and has the capability to propagate uncertainties to the output label space by considering the constraints imposed by the random projection matrix. There are several benefits of the proposed method: 1) first it naturally handles missing data by marginalizing over the unobserved labels, 2) the Bayesian perspective leads to valid probabilities that reflect the true uncertainties in the system, which in turn helps guide active learning procedures, 3) finally, the experiments show that the model significantly outperforms state-of-the-art compressed sensing based multilabel classification methods.

## 2.1 A Model for Multilabel Classification with Bayesian Compressed Sensing

We propose a model that simultaneously handles two key aspects: first is the task of compressing and recovering the label vector $\mathbf{y}_i$ to and from the lower dimensional representation $\mathbf{z}_i$. Second, given an input data $\mathbf{x}_i$ the problem is estimating low dimensional representation in the compressed space. Instead of separately solving each of the tasks, the proposed approach aims at achieving better performance by considering both of these tasks jointly, thereby modeling statistical relationships amongst different variables of interest.

Figure 1 illustrates the factor graph corresponding to the proposed model. For every data point $\mathbf{x}_i$, the output labels $\mathbf{y}_i$ influence the compressed latent vector $\mathbf{z}_i$ via the random projection matrix $\boldsymbol{\Phi}$. These compressed signals in turn also get influenced by the $d$-dimensional feature vector $\mathbf{x}_i$ via the $K$ different linear regression functions represented as a $d \times K$ matrix $\mathbf{W}$. Consequently, the role of $\mathbf{z}_i$ is not only to compress the output space but also to consider the compatibility with the input data point. The latent variable $\mathbf{W}$ corresponding to the linear model has a spherical Gaussian prior and is motivated by Gaussian Process regression [19]. Note that when $\mathbf{z}_i$ is observed, the model reduces to simple Gaussian Process regression.

One of the critical assumptions in compressed sensing is that the output labels $\mathbf{y}_i$ is sparse. The proposed model induces this constraint via a zero-mean Gaussian prior on each of the labels (i.e. $y_i^j \sim N(0, 1/\alpha_i^j)$), where the precision $\alpha_i^j$ of the normal distribution follows a Gamma prior $\alpha_i^j \sim \Gamma(a^0, b^0)$ with hyper-parameters $a^0$ and $b^0$. The Gamma prior has been earlier proposed in the context of Relevance Vector Machine (RVM) [20] as it not only induces sparsity but also is a conjugate prior to the precision $\alpha_i^j$ of the zero mean Gaussian distributions. Intuitively, marginalizing the precision in the product of Gamma priors and the Gaussian likelihoods leads to a potential function on the labels that is a student-t distribution and has a significant probability mass around zero. Thus, the labels $y_i^j$ naturally tend to zero unless they need to explain observed data. Finally, the conjugate-exponential form between the precisions $\boldsymbol{\alpha}_i$ and the output labels $\mathbf{y}_i$ leads to an efficient inference procedure that we describe later in the paper.

Note that, for labeled training data $\mathbf{x}_i \in \mathbf{X}_{\mathcal{L}}$ all the labels $\mathbf{y}_i$ are observed, while only some or none of the labels are observed for the partially labeled and test cases respectively. The proposed model ties the input feature vectors $\mathbf{X}$ to the output space $\mathbf{Y}$ via the compressed representations $\mathbf{Z}$ according to the following distribution:

$$p(\mathbf{Y}, \mathbf{Z}, \mathbf{W}, [\boldsymbol{\alpha}_i]_{i=1}^N | \mathbf{X}, \boldsymbol{\Phi}) = \frac{1}{Z} p(\mathbf{W}) \prod_{i=1}^N f_{\mathbf{x}_i}(\mathbf{w}, \mathbf{z}_i) g_{\boldsymbol{\Phi}}(\mathbf{y}_i, \mathbf{z}_i) h_{\boldsymbol{\alpha}_i}(\mathbf{y}_i) p(\boldsymbol{\alpha}_i)$$

where $Z$ is the partition function (normalization term), $p(\mathbf{W}) = \prod_{i=1}^K N(\mathbf{w}_i, 0, I)$ is the spherical Gaussian prior on the linear regression functions and $p(\boldsymbol{\alpha}_i) = \prod_{j=1}^L \Gamma(\alpha_i^j; a_0, b_0)$ is the product of Gamma priors on each individual label. Finally, the potentials $f_{\mathbf{x}_i}(\cdot, \cdot), g_{\boldsymbol{\Phi}}(\cdot, \cdot)$ and $h_{\boldsymbol{\alpha}_i}(\cdot)$ take the

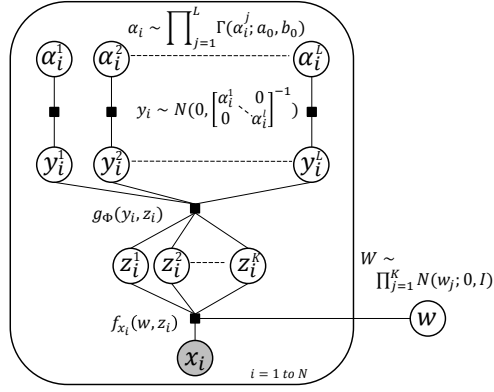

Figure 1: A Bayesian model for multilabel classification via compressed sensing. The input data is $\mathbf{x}_i$ with multiple labels $\mathbf{y}_i$, which are fully observed for the case of fully labeled training data set $\mathcal{L}$, partially observed for training data with missing labels $\mathcal{P}$, or completely unobserved as in test data $\mathcal{U}$. The latent variables $\mathbf{z}_i$ indicate the compressed label space, and $\boldsymbol{\alpha}_i$ with independent Gamma priors enforce the sparsity. The set of regression functions described by $\mathbf{W}$ is also a latent random variable and is connected across all the data points.

following form:

$$f_{\mathbf{x}_i}(\mathbf{W}, \mathbf{z}_i) = e^{-\frac{||\mathbf{W}^T\mathbf{x}_i - \mathbf{z}_i||^2}{2\sigma^2}}, \quad g_{\boldsymbol{\Phi}}(\mathbf{y}_i, \mathbf{z}_i) = e^{-\frac{||\boldsymbol{\Phi}\mathbf{y}_i - \mathbf{z}_i||^2}{2\chi^2}}, \quad h_{\boldsymbol{\alpha}_i}(\mathbf{y}_i) = \prod_{j=1}^{L} N(y_i^j; 0, \frac{1}{\alpha_i^j}).$$

Intuitively, the potential term $f_{\mathbf{x}_i}(\mathbf{W}, \mathbf{z}_i)$ favors configurations that are aligned with output of the linear regression function when applied to the input feature vector. Similarly, the term $g_{\boldsymbol{\Phi}}(\mathbf{y}_i, \mathbf{z}_i)$ favors configurations that are compatible with the output compressive projections determined by $\boldsymbol{\Phi}$. Finally, as described earlier, $h_{\boldsymbol{\alpha}_i}(\mathbf{y}_i)$ enforces sparsity in the output space. The parameters $\sigma^2$ and $\chi^2$ denote noise parameters and determine how tight the relation is between the labels in the output space, the compressed space and the regression coefficients. By changing the value of these parameters we can emphasize or de-emphasize the relationship between the latent variables.

In summary, our model provides a powerful framework for modeling multilabel classification using compressive sensing. The model promises statistical efficiency by jointly considering compressive sensing and regression within a single model. Moreover, as we will see in the next section this model allows efficient numerical procedures for inferring unobserved labels by resolving the constraints imposed by the potential functions and the observed data. The model naturally handles the case of missing data (incomplete labels) by automatically marginalizing the unobserved data as a part of the inference mechanism. Finally, the probabilistic nature of the approach provides us with valid probabilistic quantities that can be used to perform active selection of the unlabeled points.

## 2.2 Inference

First, consider the simpler scenario where the training data set only consists of fully labeled instances $\mathbf{X}_{\mathcal{L}}$ with labels $\mathbf{Y}_{\mathcal{L}}$. Thus our aim is to infer $p(\mathbf{Y}_{\mathcal{U}}|\mathbf{X}, \mathbf{Y}_{\mathcal{L}}, \boldsymbol{\Phi})$, the posterior distribution over unlabeled data. Performing exact inference is prohibitive in this model primarily due to the following reason. First, notice that the joint distribution is a product of a Gaussian (Spherical prior on $\mathbf{W}$ and compatibility terms with $\mathbf{z}_i$) and non-Gaussian terms (the Gamma priors). Along with these sparsity terms, the projection of the label space into the compressed space precludes usage of exact inference via a junction tree algorithm. Thus, we resort to approximate inference techniques. In particular we perform an approximate inference by maximizing the variational lower bound by assuming that the posterior over the unobserved random variable $\mathbf{W}$, $\mathbf{Y}_{\mathcal{U}}$, $\mathbf{Z}$ and $[\boldsymbol{\alpha}_i]_{i=1}^N$ can be factorized:

$$F = \int_{\mathbf{Y}_{\mathcal{U}}, \mathbf{Z}, \mathbf{W}, [\boldsymbol{\alpha}]_{i=1}^N} q(\mathbf{Y}_{\mathcal{U}})q(\mathbf{Z})q(\mathbf{W})q([\boldsymbol{\alpha}_i]_{i=1}^N) \log \frac{p(\mathbf{Y}, \mathbf{Z}, \mathbf{W}, [\boldsymbol{\alpha}_i]_{i=1}^N|\mathbf{X}, \boldsymbol{\Phi})}{q(\mathbf{Y}_{\mathcal{U}})q(\mathbf{Z})q(\mathbf{W})q([\boldsymbol{\alpha}_i]_{i=1}^N)}$$

$$\leq \log \int_{\mathbf{Y}_{\mathcal{U}}, \mathbf{Z}, \mathbf{W}, [\boldsymbol{\alpha}]_{i=1}^N} p(\mathbf{Y}, \mathbf{Z}, \mathbf{W}, [\boldsymbol{\alpha}_i]_{i=1}^N|\mathbf{X}, \boldsymbol{\Phi})$$

Here, the posteriors on the precisions $\boldsymbol{\alpha}_i$ are assumed to be Gamma distributed while the rest of the distributions are constrained to be Gaussian. Further, each of these joint posterior densities are assumed to have the following per data point factorization: $q(\mathbf{Y}_{\mathcal{U}}) = \prod_{i \in \mathcal{U}} q(\mathbf{y}_i)$, $q(\mathbf{Z}) = \prod_{i \in \mathcal{U} \cup \mathcal{L}} q(\mathbf{z}_i)$ and $q([\boldsymbol{\alpha}]_{i=1}^N) = \prod_{i=1}^N \prod_{j=1}^l q(\alpha_i^j)$. Similarly the posterior over the regression functions has a per dimension factorization: $q(\mathbf{W}) = \prod_{i=1}^K q(\mathbf{w}_i)$. The approximate inference algorithm aims to compute good approximations to the real posteriors by iteratively optimizing the above described variational bound. Specifically, given the approximations $q^t(\mathbf{y}_i) \sim N(\mu_{\mathbf{y}_i}^t, \Sigma_{\mathbf{y}_i}^t)$ (similar forms for $\mathbf{z}_i$ and $\mathbf{w}_i$) and $q^t(\alpha_i^j) \sim \Gamma(a_{ij}^t, b_{ij}^t)$ from the $t^{th}$ iteration the update rules are as follows:

Update for $q^{t+1}(\mathbf{y}_i)$: $\Sigma_{\mathbf{y}_i}^{t+1} = [\text{diag}(\mathbb{E}(\boldsymbol{\alpha}_i^t)) + \Phi^T \chi^{-2} \Phi]^{-1}$,         $\mu_{\mathbf{y}_i}^{t+1} = \Sigma_{\mathbf{y}_i}^{t+1} \Phi^T \chi^{-2} \mu_{\mathbf{z}_i}^t$,

Update for $q^{t+1}(\alpha_i^j)$: $a_{ij}^{t+1} = a_{ij}^0 + 0.5$,         $b_{ij}^{t+1} = b_{ij}^0 + 0.5[\Sigma_{\mathbf{y}_i}^{t+1}(j,j) + [\mu_{\mathbf{y}_i}^{t+1}(j)]^2]$,

Update for $q^{t+1}(\mathbf{z}_i)$: $\Sigma_{\mathbf{z}_i}^{t+1} = [\sigma^{-2} I + \chi^{-2} I]^{-1}$,         $\mu_{\mathbf{z}_i}^{t+1} = \Sigma_{\mathbf{z}_i}^{t+1}[\sigma^{-2}[\mu_{\mathbf{W}}^{t+1}]^T \mathbf{x}_i + \chi^{-2} \Phi \mu_{\mathbf{y}_i}^{t+1}]$,

Update for $q^{t+1}(\mathbf{w}_i)$: $\Sigma_{\mathbf{w}_i}^{t+1} = [\sigma^{-2} \mathbf{X}\mathbf{X}^T + I]^{-1}$,         $\mu_{\mathbf{w}_i}^{t+1} = \sigma^{-2} \Sigma_{\mathbf{w}_i}^{t+1} \mathbf{X}[\mu_{\mathbf{z}}^{t+1}(i)]^T$.

Alternating between the above described updates can be considered as message passing between the low-dimensional regression outputs and higher dimensional output labels, which in turn are constrained to be sparse. By doing the update on $q(\mathbf{y}_i)$, the algorithms attempts to explain the compressed signal $\mathbf{z}_i$ using sparsity imposed by the precisions $\boldsymbol{\alpha}_i$. Similarly, by updating $q(\mathbf{z}_i)$ and $q(\mathbf{W})$ the inference procedures reasons about a compressed representation that is most efficient in terms of reconstruction. By iterating between these updates the model consolidates information from the two key components, compressed sensing and regression, that constitute the system and is more effective than doing these tasks in isolation.

Also note that the most expensive step is in the first update for computing $\Sigma_{y_i}^{t+1}$ , which if naively implemented would require an inversion of an $L \times L$ matrix. However, this inversion can be computed easily using Sherman-Morrison-Woodbury formula, which in turn reduces the complexity of the update to $O(K^3 + K^2 L)$. The only other significant update is the posterior computation $q(\mathbf{w})$ that is $O(d^3)$, where $d$ is the dimensionality of the feature space. Consequently, this scheme is fairly efficient and has time complexity similar to that of other non-probabilistic approaches. Finally, note that straightforward extension to non-linear regression functions can be done via the kernel trick.

**Handling Missing Labels in Training Data:** The proposed model and the inference procedure naturally handles the case of missing labels in the training set via the variational inference. Lets consider a data point $\mathbf{x}_p$ with set of partially observed labels $\mathbf{y}_p^o$. If we denote $\mathbf{y}_p^u$ as the set of unobserved labels, then all the above mentioned update steps stay the same except for the one that updates $q(\mathbf{z}_p)$, which takes the following form:

$$\mu_{\mathbf{z}_p}^{t+1} = \Sigma_{\mathbf{z}_p}^{t+1}[\sigma^{-2}\mathbf{x}_p^T \mu_{\mathbf{W}}^{t+1} + \chi^{-2} \Phi^{uo}[\mu_{\mathbf{y}_p^u}^{t+1}; \mathbf{y}_p^o]].$$

Here $\Phi^{uo}$ denotes re-ordering of the columns on $\Phi$ according to the indices of the observed and unobserved labels. Intuitively, the compressed signal $\mathbf{z}_p$ now considers compatibility with the unobserved labels, while taking into account the observed labels, and in doing so effectively facilitates message passing between all the latent random variables.

**Handling a Test Point:** While it might seem that the above mentioned framework works in the transductive setting, we here show such is not the case and that the framework can seamlessly handle test data in an inductive setting. Note that given a training set, we can recover the posterior distribution $q(\mathbf{W})$ that summarizes the regression parameter. This posterior distribution is sufficient for doing inference on a test point $\mathbf{x}_*$. Intuitively, the key idea is that the information about the training set is fully captured in the regression parameters, thus, the labels for the test point can be simply recovered by only iteratively updating $q(\mathbf{y}_*)$, $q(\mathbf{z}_*)$ and $q(\boldsymbol{\alpha}_*)$.

## 2.3 Active Learning

The main aim in active learning is to seek bits of information that would promise to enhance the discriminatory power of the framework the most. When employed in a traditional classification setting, the active learning procedure boils down to the task of seeking the label for one of the unlabeled examples that promises to be most informative and then update the classification model by incorporating it into the existing training set. However, multilabel classification enables richer forms of active information acquisitions, which we describe below:

- **Traditional Active Learning:** This is similar to the active learning scenario in traditional classification tasks. In particular, the goal is to select an unlabeled sample for which *all the labels will be revealed.*

- **Active Diagnosis:** Given a test data point, at every iteration the active acquisition procedure seeks a label for each test point that is maximally informative for the same and promises to improve the prediction accuracy over the rest of the unknown labels.

Note that *Active Diagnosis* is highly relevant for real-world tasks. For example, consider the wikipedia page classification problem. Just knowing a few labels about the page can be immensely useful in inferring the rest of the labels. Active diagnosis should be able to leverage the statistical dependency amongst the output label space, in order to ask for labels that are maximally informative.

A direct generalization of the above two paradigms is a setting in which the active learning procedure selects a label for one point in the training set. Specifically, the key difference between this scenario and the traditional active learning is that only one label is chosen to be revealed for the selected data point instead of the entire set of labels.

Non-probabilistic classification schemes, such as SVMs, can handle traditional active learning by first establishing the confidence in the estimate of each label by using the distance from the classification boundary (margin) and then selecting the point that is closest to the margin. However, it is fairly non-trivial to extend those approaches to tackle the active diagnosis and generalized information acquisition. On the other hand the proposed Bayesian model provides a posterior distribution over the unknown class labels as well as other latent variables and can be used for active learning.

In particular, measures such as uncertainty or information gain can be used to guide the selective sampling procedure for active learning. Formally, we can write these two selection criteria as:

$$\text{Uncertainty: } \arg\max_{y_i \in \mathbf{Y}_{\mathcal{U}}} H(y_i)$$
$$\text{InfoGain: } \arg\max_{y_i \in \mathbf{Y}_{\mathcal{U}}} H(\mathbf{Y}_{\mathcal{U}}/y_i) - \mathbb{E}_{y_i}[H(\mathbf{Y}_{\mathcal{U}}/y_i|y_i)].$$

Here, $H(\cdot)$ denotes Shannon entropy and is a measure of uncertainty. The uncertainty criterion seeks to select the labels that have the highest entropy, whereas the information gain criterion seeks to select a label that has the highest expected reduction in uncertainty over all the other unlabeled points or unknown labels. Either of these criteria can be computed given the inferred posteriors; however we note that the information gain criterion is far more expensive to compute as it requires repeated inference by considering all possible labels for every unlabeled data point. The uncertainty criterion on the other hand is very simple and often guides active learning with reasonable amount of gains. In this work we will consider uncertainty as the primary active learning criterion. Finally, we'd like to point that the different described forms of active learning can naturally be addressed with these heuristics by appropriately choosing the set of possible candidates and the posterior distributions over which the entropy is measured.

## 3 Experiments

In this section, we present experimental results using our methods on standard benchmark datasets. The goals of our experiments are three-fold: a) demonstrate that the proposed jointly probabilistic method is significantly better than the standard compressed sensing based method by [1] and gets comparable accuracy to 1-vs-all SVM while projecting labels onto much smaller dimensionality $K$ compared to the total number of labels $L$, b) show robustness of our method to missing labels, c) demonstrate various active learning scenarios and compare them against the standard baselines. We use Matlab for all our implementations. We refer to our Compressed Sensing based Bayesian Multilabel classification method as BML-CS . In BML-CS method, the hyper-parameters $a^0$ and $b^0$ are set to $10^{-6}$, which in turn leads to a fairly uninformative prior. The noise parameters $\chi$ and $\sigma$ are found by maximizing the marginalized likelihood of the Gaussian Process Regression model [19]. We use liblinear for SVM implementation; error penalty $C$ is selected using cross-validation. We also implemented the multilabel classification method based on compressed sensing (ML-CS ) [1] with CoSamp [8] being the underlying sparse vector recovery algorithm.

For our experiments, we use standard multilabel datasets. In particular, we choose datasets where the number of labels is high. Such datasets generally tend to have only a few labels per data point and the compressed sensing methods can exploit this sparsity to their advantage.

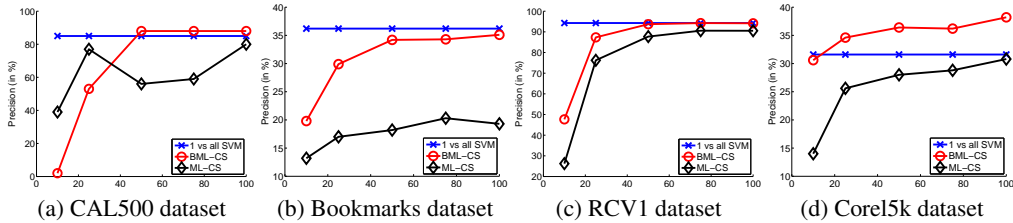

| (a) CAL500 dataset | (b) Bookmarks dataset | (c) RCV1 dataset | (d) Corel5k dataset |

Figure 2: Comparison of precision values (in top-1 label) for different methods with different values of $K$, dimensionality of the compressed label space. The SVM baseline uses all the $L$ labels. The x-axis shows $K$ as a percentage of the total number of labels $L$. Clearly, for each of the dataset the proposed method obtains accuracy similar to 1-vs-all SVM method while projecting to only $K = L/2$ dimensions. Also, our method consistently obtains significantly higher accuracies than the CS method of [1] while using the same number of latent variables $K$.

| $K$ | Top-3 SVM | Top-3 BML-CS | Top-3 ML-CS | Top-5 SVM | Top-5 BML-CS | Top-5 ML-CS |
|---|---|---|---|---|---|---|
| 10% | | 0.04 | 0.36 | | 0.09 | 0.32 |
| 25% | | 0.38 | 0.48 | | 0.28 | 0.41 |
| 50% | 0.74 | 0.61 | 0.44 | 0.67 | 0.51 | 0.40 |
| 75% | | 0.75 | 0.53 | | 0.60 | 0.55 |
| 100% | | 0.70 | 0.61 | | 0.65 | 0.57 |

**(a)**

| $K$ | Top-3 SVM | Top-3 BML-CS | Top-3 ML-CS | Top-5 SVM | Top-5 BML-CS | Top-5 ML-CS |
|---|---|---|---|---|---|---|
| 10% | | 0.10 | 0.06 | | 0.07 | 0.04 |
| 25% | | 0.15 | 0.08 | | 0.10 | 0.05 |
| 50% | 0.20 | 0.17 | 0.09 | 0.14 | 0.12 | 0.06 |
| 75% | | 0.17 | 0.10 | | 0.13 | 0.07 |
| 100% | | 0.19 | 0.10 | | 0.13 | 0.07 |

**(b)**

| $K$ | Top-3 SVM | Top-3 BML-CS | Top-3 ML-CS | Top-5 SVM | Top-5 BML-CS | Top-5 ML-CS |
|---|---|---|---|---|---|---|
| 10% | | 0.33 | 0.19 | | 0.23 | 0.14 |
| 25% | | 0.65 | 0.59 | | 0.44 | 0.39 |
| 50% | 0.75 | 0.75 | 0.69 | 0.54 | 0.52 | 0.49 |
| 75% | | 0.75 | 0.71 | | 0.53 | 0.50 |
| 100% | | 0.75 | 0.72 | | 0.53 | 0.51 |

**(c)**

| $K$ | Top-3 SVM | Top-3 BML-CS | Top-3 ML-CS | Top-5 SVM | Top-5 BML-CS | Top-5 ML-CS |
|---|---|---|---|---|---|---|
| 10% | | 0.20 | 0.08 | | 0.15 | 0.06 |
| 25% | | 0.27 | 0.17 | | 0.22 | 0.14 |
| 50% | 0.27 | 0.27 | 0.21 | 0.22 | 0.23 | 0.17 |
| 75% | | 0.27 | 0.22 | | 0.23 | 0.18 |
| 100% | | 0.27 | 0.22 | | 0.23 | 0.17 |

**(d)**

Figure 3: Precision values obtained by various methods in retrieving 3 and 5 labels respectively. First column in each table shows $K$ as the fraction of number of labels $L$. 1-vs-all SVM requires training $L$ weight vectors, while both BML-CS and ML-CS trains $K$ weight vectors. BML-CS is consistently more accurate than ML-CS although its accuracy is not as close to SVM as it is for the case of top-1 labels (see Figure 2).

For each of the algorithms we recover the top 1, 3, 5 most likely positive labels and set remaining labels to be negative. For each value of $t \in \{1, 3, 5\}$, we report precision in prediction, i.e., fraction of true positives to the total number of positives predicted.

### 3.1 Multilabel Classification Accuracies

We train both ML-CS and our method BML-CS on all datasets using different values of $K$, i.e., the dimensionality of the space of latent variables $\mathbf{z}$ for which weight vectors are learned. Figure 2 compares precision (in predicting 1 positive label) of our proposed method on four different datasets for different values of $K$ with the corresponding values obtained by ML-CS and SVM . Note that 1-vs-all SVM learns all $L > K$ weight vectors, hence it is just one point in the plot; we provide a line for ease of comparison. It is clear from the figure that both BML-CS and ML-CS are significantly worse than 1-vs-all SVM when $K$ is very small compared to total number of labels $L$. However, for around $K = 0.5L$, our method achieves close to the baseline (1-vs-all SVM) accuracy while ML-CS still achieves significantly worse accuracies. In fact, even with $K = L$, ML-CS still obtains significantly lower accuracy than SVM baseline.

In Figure 3 we tabulate precision for top-3 and top-5 retrieved positive labels. Here again, the proposed method is consistently more accurate than ML-CS . However, it requires larger $K$ to obtain similar precision values as SVM. This is fairly intuitive as for higher recall rate the multilabel problems become harder and hence our method requires more weight vectors to be learned per label.

### 3.2 Missing Labels

Next, we conduct experiments for multilabel classification with missing labels. Specifically, we remove a fixed fraction of training labels randomly from each dataset considered. We then apply

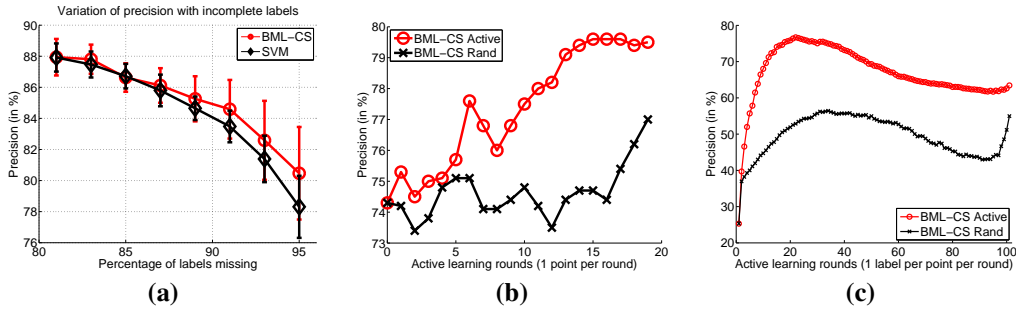

**(a)**          **(b)**          **(c)**

Figure 4: **(a)** Precision (in retrieving the most likely positive label) obtained by BML-CS and SVM methods on RCV1 dataset with varying fraction of missing labels. We observe that BML-CS obtains higher precision values than baseline SVM.($k = 0.5L$) **(b)** Precision obtained after each round of active learning by BML-CS-Active method and by the baseline random selection strategy over RCV1 dataset.**(c)** Precision after active learning, where one label per point is added to the training set, in comparison with random baseline on RCV1 dataset. Parameters for (b) & (c): $k = 0.1L$. Both (b) and (c), start with 100 points initially.

BML-CS as well as 1-vs-all SVM method to such training data. Since, SVM cannot directly handle missing labels, we always set a missing label to be a negative label. In contrast, our method can explicitly handle missing labels and can perform inference by marginalizing the unobserved tags.

As the number of positive labels is significantly smaller than the negative labels, when only a small fraction of labels are removed, both SVM and BML-CS obtain similar accuracies to the case where all the labels are present. However, as the number of missing labels increase there is a smooth dip in the precision of the two methods. Figure 4 (a) compares precision obtained by BML-CS with the precision obtained by 1-vs-all SVM. Clearly, our method performs better than SVM, while using only $K = .5L$ weight vectors.

### 3.3 Active Learning

In this section, we provide empirical results for some of the active learning tasks we discussed in Section 2.3. For each of the tasks, we use our Bayesian multilabel method to compute the posterior over the label vector. We then select the desired label/point appropriately according to each individual task. For each of the tasks, we compare our method against an appropriate baseline method.

**Traditional Active Learning:** The goal here is to select most informative points which if labeled completely will increase the accuracy by the highest amount. We use uncertainty sampling where we consider the entropy of the posterior over label vector as the selection criterion for BML-CS-Active method. We compare the proposed method against the standard random selection baseline. For these experiments, we initialize both the methods with an initial labeled dataset of 100 points and then after each active learning round we seek all the labels for the selected training data point. Figure 4 (b) compares precisions obtained by BML-CS-Active method with the precisions obtained by the baseline method after every active learning round. After just 15 active learning rounds, our method is able to gain about 6% of accuracy while random selection method do not provide any gain in the accuracy.

**Active Diagnosis:** In this type of active learning, we query one label for each of the training points in each round. For each training point, we choose a label with the most uncertainty and ask for its label. Figure 4 (c) plots the improvement in precision values with number of rounds of active learning, for estimating the top-1 label. From the plot, we can see that after just 20 rounds, choosing points by uncertainty has an improvement of 20% over the random baseline.

## 4 Conclusion and Future Work

We presented a Bayesian framework for multilabel classification that uses compressive sensing. The proposed framework jointly models the compressive sensing/reconstruction task with learning regression over the compressed space. We present an efficient variational inference scheme that jointly resolves compressed sensing and regression tasks. The resulting posterior distribution can further be used to perform different flavors of active learning. Experimental evaluations highlight the efficacy of the framework. Future directions include considering other structured prediction tasks that are sparse and applying the framework to novel scenarios. Further, instead of myopic next best information seeking we also seek to investigate non-myopic selective sampling where an optimal subset of unlabeled data are selected.

# References

[1] D. Hsu, S. Kakade, J. Langford, and T. Zhang. Multi-label prediction via compressed sensing. In *NIPS*, pages 772–780, 2009.

[2] B. Hariharan, L. Zelnik-Manor, S. V. N. Vishwanathan, and M. Varma. Large scale max-margin multi-label classification with priors. In *ICML*, pages 423–430, 2010.

[3] G. Tsoumakas and I. Katakis. Multi-label classification: An overview. *IJDWM*, 3(3):1–13, 2007.

[4] I. Tsochantaridis, T. Joachims, T. Hofmann, and Y. Altun. Large margin methods for structured and interdependent output variables. *Journal of Machine Learning Research*, 6:1453–1484, 2005.

[5] M. R. Boutell, J. Luo, X. Shen, and C. M. Brown. Learning multi-label scene classification. *Pattern Recognition*, 37(9):1757–1771, 2004.

[6] B. Taskar, C. Guestrin, and D. Koller. Max-margin markov networks. In *NIPS*, 2003.

[7] R. M. Rifkin and A. Klautau. In defense of one-vs-all classification. *Journal of Machine Learning Research*, 5:101–141, 2004.

[8] D. Needell and J. A. Tropp. Cosamp: Iterative signal recovery from incomplete and inaccurate samples. *Applied and Computational Harmonic Analysis*, 26(3):301 – 321, 2009.

[9] S. Foucart. Hard thresholding pursuit: an algorithm for compressive sensing, 2010. preprint.

[10] D. Baron, S. S. Sarvotham, and R. G. Baraniuk. Bayesian compressive sensing via belief propagation. *IEEE Transactions on Signal Processing*, 58(1), 2010.

[11] S. Ji, Y. Xue, and L. Carin. Bayesian compressive sensing. *IEEE Transactions on Signal Processing*, 56(6), 2008.

[12] N. Cesa-Bianchi, A Conconi, and C. Gentile. Learning probabilistic linear-threshold classifiers via selective sampling. In *COLT*, 2003.

[13] N. Lawrence, M. Seeger, and R. Herbrich. Fast sparse Gaussian Process method: Informative vector machines. *NIPS*, 2002.

[14] D. MacKay. Information-based objective functions for active data selection. *Neural Computation*, 4(4), 1992.

[15] S. Tong and D. Koller. Support vector machine active learning with applications to text classification. In *ICML*, 2000.

[16] Y. Freund, H. S. Seung, E. Shamir, and N. Tishby. Selective sampling using the query by committee algorithm. *Machine Learning*, 28(2-3), 1997.

[17] B. Yang, J.-Tao Sun, T. Wang, and Z. Chen. Effective multi-label active learning for text classification. In *KDD*, pages 917–926, 2009.

[18] J. Weston, S. Bengio, and N. Usunier. Large scale image annotation: learning to rank with joint word-image embeddings. *Machine Learning*, 81(1):21–35, 2010.

[19] C. E. Rasmussen and C. K. I. Williams. *Gaussian Processes for Machine Learning (Adaptive Computation and Machine Learning)*. The MIT Press, 2005.

[20] M. E. Tipping. Sparse bayesian learning and the relevance vector machine. *Journal of Machine Learning Research*, 1:211–244, 2001.

